# Learning sparse codes with a mixture-of-Gaussians prior

**Bruno A. Olshausen**
Department of Psychology and
Center for Neuroscience, UC Davis
1544 Newton Ct.
Davis, CA 95616
*baolshausen@ucdavis.edu*

**K. Jarrod Millman**
Center for Neuroscience, UC Davis
1544 Newton Ct.
Davis, CA 95616
*kjmillman@ucdavis.edu*

## Abstract

We describe a method for learning an overcomplete set of basis functions for the purpose of modeling sparse structure in images. The sparsity of the basis function coefficients is modeled with a mixture-of-Gaussians distribution. One Gaussian captures non-active coefficients with a small-variance distribution centered at zero, while one or more other Gaussians capture active coefficients with a large-variance distribution. We show that when the prior is in such a form, there exist efficient methods for learning the basis functions as well as the parameters of the prior. The performance of the algorithm is demonstrated on a number of test cases and also on natural images. The basis functions learned on natural images are similar to those obtained with other methods, but the sparse form of the coefficient distribution is much better described. Also, since the parameters of the prior are adapted to the data, no assumption about sparse structure in the images need be made *a priori*, rather it is learned from the data.

## 1 Introduction

The general problem we address here is that of learning a set of basis functions for representing natural images efficiently. Previous work using a variety of optimization schemes has established that the basis functions which best code natural images in terms of sparse, independent components resemble a Gabor-like wavelet basis in which the basis functions are spatially localized, oriented and bandpass in spatial-frequency [1, 2, 3, 4]. In order to tile the joint space of position, orientation, and spatial-frequency in a manner that yields useful image representations, it has also been advocated that the basis set be *overcomplete* [5], where the number of basis functions exceeds the dimensionality of the images being coded. A major challenge in learning overcomplete bases, though, comes from the fact that the posterior distribution over the coefficients must be sampled during learning. When the posterior is sharply peaked, as it is when a sparse prior is imposed, then conventional sampling methods become especially cumbersome.

One approach to dealing with the problems associated with overcomplete codes and sparse priors is suggested by the form of the resulting posterior distribution over the coefficients averaged over many images. Shown below is the posterior distribution of one of the coefficients in a 4×'s overcomplete representation. The sparse prior that was imposed in learning was a Cauchy distribution and is overlaid (dashed line). It would seem that the coefficients do not fit this imposed prior very well, and instead want to occupy one of two states: an *inactive* state in which the coefficient is set nearly to zero, and an *active* state in which the coefficient takes on some significant non-zero value along a continuum. This suggests that the appropriate choice of prior is one that is capable of capturing these two discrete states.

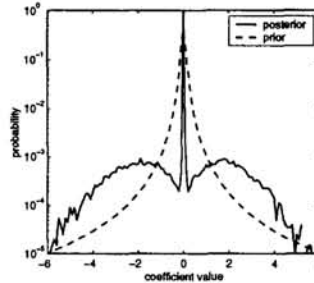

Figure 1: Posterior distribution of coefficients with Cauchy prior overlaid.

Our approach to modeling this form of sparse structure uses a *mixture-of-Gaussians* prior over the coefficients. A set of binary or ternary state variables determine whether the coefficient is in the active or inactive state, and then the coefficient distribution is Gaussian distributed with a variance and mean that depends on the state variable. An important advantage of this approach, with regard to the sampling problems mentioned above, is that the use of Gaussian distributions allows an analytical solution for integrating over the posterior distribution for a given setting of the state variables. The only sampling that needs to be done then is over the binary or ternary state variables. We show here that this problem is a tractable one. This approach differs from that taken previously by Attias [6] in that we do not use variational methods to approximate the posterior, but rather we rely on sampling to adequately characterize the posterior distribution over the coefficients.

## 2    Mixture-of-Gaussians model

An image, $I(x,y)$, is modeled as a linear superposition of basis functions, $\phi_i(x,y)$, with coefficients $a_i$, plus Gaussian noise $\nu(x,y)$:

$$I(x,y) = \sum_i a_i\,\phi_i(x,y) + \nu(x,y) \qquad (1)$$

In what follows this will be expressed in vector-matrix notation as $\mathbf{I} = \mathbf{\Phi}\,\mathbf{a} + \nu$.

The prior probability distribution over the coefficients is factorial, with the distribution over each coefficient $a_i$ modeled as a mixture-of-Gaussians distribution with either two or three Gaussians (fig. 2). A set of binary or ternary state variables $s_i$ then determine which Gaussian is used to describe the coefficients.

The total prior over both sets of variables, $\mathbf{a}$ and $\mathbf{s}$, is of the form

$$P(\mathbf{a},\mathbf{s}) = \prod_i P(a_i|s_i)\,P(s_i) \qquad (2)$$

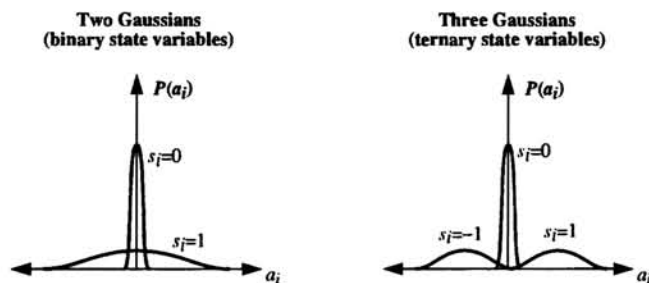

Figure 2: Mixture-of-Gaussians prior.

where $P(s_i)$ determines the probability of being in the active or inactive states, and $P(a_i|s_i)$ is a Gaussian distribution whose mean and variance is determined by the current state $s_i$.

The total image probability is then given by

$$P(\mathbf{I}|\theta) = \sum_{\mathbf{s}} P(\mathbf{s}|\theta) \int P(\mathbf{I}|\mathbf{a}, \theta) P(\mathbf{a}|\mathbf{s}, \theta) d\mathbf{a} \tag{3}$$

where

$$P(\mathbf{I}|\mathbf{a}, \theta) = \frac{1}{Z_{\lambda_N}} e^{-\frac{\lambda_N}{2}|\mathbf{I} - \Phi \mathbf{a}|^2} \tag{4}$$

$$P(\mathbf{a}|\mathbf{s}, \theta) = \frac{1}{Z_{\Lambda_{\mathbf{a}}(\mathbf{s})}} e^{-\frac{1}{2}(\mathbf{a} - \mu(\mathbf{s}))^t \Lambda_{\mathbf{a}}(\mathbf{s})(\mathbf{a} - \mu(\mathbf{s}))} \tag{5}$$

$$P(\mathbf{s}|\theta) = \frac{1}{Z_{\Lambda_{\mathbf{s}}}} e^{-\frac{1}{2}\mathbf{s}^t \Lambda_{\mathbf{s}} \mathbf{s}} \tag{6}$$

and the parameters $\theta$ include $\lambda_N$, $\Phi$, $\Lambda_{\mathbf{a}}(\mathbf{s})$, $\mu(\mathbf{s})$, and $\Lambda_{\mathbf{s}}$. $\Lambda_{\mathbf{a}}(\mathbf{s})$ is a diagonal inverse covariance matrix with elements $\Lambda_{\mathbf{a}}(\mathbf{s})_{ii} = \lambda_{a_i}(s_i)$. (The notations $\Lambda_{\mathbf{a}}(\mathbf{s})$ and $\mu(\mathbf{s})$ are used here to explicitly reflect the dependence of the means and variances of the $a_i$ on $s_i$.) $\Lambda_{\mathbf{s}}$ is also diagonal (for now) with elements $\Lambda_{\mathbf{s}ii} = \lambda_{s_i}$. The model is illustrated graphically in figure 3.

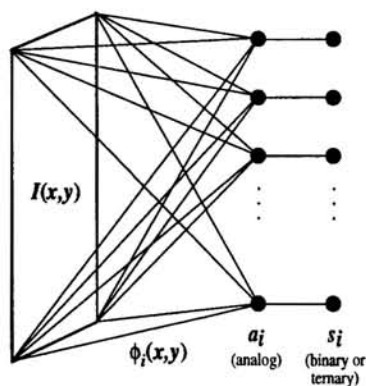

Figure 3: Image model.

## 3 Learning

The objective function for learning the parameters of the model is the average log-likelihood:

$$\mathcal{L} = \langle \log P(\mathbf{I}|\theta) \rangle \tag{7}$$

Maximizing this objective will minimize the lower bound on coding length.

Learning is accomplished via gradient ascent on the objective, $\mathcal{L}$. The learning rules for the parameters $\boldsymbol{\Lambda_s}$, $\boldsymbol{\Lambda_a}(\mathbf{s})$, $\mu(\mathbf{s})$ and $\boldsymbol{\Phi}$ are given by:

$$\Delta\lambda_{s_i} \propto \frac{\partial \mathcal{L}}{\partial \lambda_{s_i}}$$

$$= \frac{1}{2}\left[\langle s_i \rangle_{P(s_i|\theta)} - \langle s_i \rangle_{P(\mathbf{s}|\mathbf{I},\theta)}\right] \tag{8}$$

$$\Delta\lambda_{a_i}(u) \propto \frac{\partial \mathcal{L}}{\partial \lambda_{a_i}(u)}$$

$$= \frac{1}{2}\left[\frac{\langle \delta(s_i - u)\rangle_{P(\mathbf{s}|\mathbf{I},\theta)}}{\lambda_{a_i}(u)} - \right.$$

$$\left. \langle \delta(s_i - u)\left(K_{ii}(u) - 2\hat{a}_i(u)\mu_i(u) + \mu_i^2(u)\right)\rangle_{P(\mathbf{s}|\mathbf{I},\theta)}\right] \tag{9}$$

$$\Delta\mu_i(u) \propto \frac{\partial \mathcal{L}}{\partial \mu_i(u)}$$

$$= \lambda_{a_i}(u)\langle \delta(s_i - u)\left(\hat{a}_i(u) - \mu_i(u)\right)\rangle \tag{10}$$

$$\Delta\boldsymbol{\Phi} \propto \frac{\partial \mathcal{L}}{\partial \boldsymbol{\Phi}}$$

$$= \lambda_N \left[\mathbf{I}\langle \hat{\mathbf{a}}(\mathbf{s})\rangle_{P(\mathbf{s}|\mathbf{I},\theta)} - \boldsymbol{\Phi}\langle \mathbf{K}(\mathbf{s})\rangle_{P(\mathbf{s}|\mathbf{I},\theta)}\right] \tag{11}$$

where $u$ takes on values 0,1 (binary) or -1,0,1 (ternary) and $\mathbf{K}(\mathbf{s}) = \mathbf{H}^{-1}(\mathbf{s}) + \hat{\mathbf{a}}(\mathbf{s})\hat{\mathbf{a}}(\mathbf{s})^T$. ($\hat{\mathbf{a}}$ and $\mathbf{H}$ are defined in eqs. 15 and 16 in the next section.) Note that in these expressions we have dropped the outer brackets averaging over images simply to reduce clutter.

Thus, for each image we must sample from the posterior $P(\mathbf{s}|\mathbf{I},\theta)$ in order to collect the appropriate statistics needed for learning. These statistics must be accumulated over many different images, and then the parameters are updated according to the rules above. Note that this approach differs from that of Attias [6] in that we do not attempt to sum over all states, $\mathbf{s}$, or to use the variational approximation to approximate the posterior. Instead, we are effectively summing only over those states that are most probable according to the posterior. We conjecture that this scheme will work in practice because the posterior has significant probability only for a small fraction of states $\mathbf{s}$, and so it can be well-characterized by a relatively small number of samples. Next we present an efficient method for Gibbs sampling from the posterior.

## 4 Sampling and inference

In order to sample from the posterior $P(\mathbf{s}|\mathbf{I},\theta)$, we first cast it in Boltzmann form:

$$P(\mathbf{s}|\mathbf{I},\theta) \propto e^{-E(\mathbf{s})} \tag{12}$$

where

$$E(\mathbf{s}) = -\log P(\mathbf{s},\mathbf{I}|\theta) = -\log P(\mathbf{s}|\theta)\int P(\mathbf{I}|\mathbf{a},\theta)P(\mathbf{a}|\mathbf{s},\theta)d\mathbf{a}$$

$$= \frac{1}{2}\mathbf{s}^T \mathbf{\Lambda_s}\,\mathbf{s} + \log Z_{\mathbf{\Lambda_a(s)}} + E_{\mathbf{a}|\mathbf{s}}(\hat{\mathbf{a}}, \mathbf{s}) + \frac{1}{2}\log \det \mathbf{H}(\mathbf{s}) + \mathrm{const.} \quad (13)$$

and

$$E_{\mathbf{a}|\mathbf{s}}(\mathbf{a}, \mathbf{s}) \equiv \frac{\lambda_N}{2}|\mathbf{I} - \mathbf{\Phi}\mathbf{a}|^2 + \frac{1}{2}(\mathbf{a} - \mu(\mathbf{s}))^T \mathbf{\Lambda_a}(\mathbf{s})\,(\mathbf{a} - \mu(\mathbf{s})) \quad (14)$$

$$\hat{\mathbf{a}} = \arg\min_{\mathbf{a}} E_{\mathbf{a}|\mathbf{s}}(\mathbf{a}, \mathbf{s}) \quad (15)$$

$$\mathbf{H}(\mathbf{s}) = \nabla\nabla_{\mathbf{a}}E_{\mathbf{a}|\mathbf{s}}(\mathbf{a}, \mathbf{s}) = \lambda_N \mathbf{\Phi}^T \mathbf{\Phi} + \mathbf{\Lambda_a}(\mathbf{s}) \quad (16)$$

Gibbs-sampling on $P(\mathbf{s}|\mathbf{I}, \theta)$ can be performed by flipping state variables $s_i$ according to

$$P(s_i \leftarrow s^\alpha) = \frac{1}{1 + e^{\Delta E(s_i \leftarrow s^\alpha)}} \qquad \text{(binary)} \quad (17)$$

$$P(s_i \leftarrow s^\alpha) = \frac{1}{1 + e^{\Delta E(s_i \leftarrow s^\alpha)}\left[1 + e^{-\Delta E(s_i \leftarrow s^\beta)}\right]} \qquad \text{(ternary)} \quad (18)$$

Where $s^\alpha = \overline{s_i}$ in the binary case, and $s^\alpha$ and $s^\beta$ are the two alternative states in the ternary case. $\Delta E(s_i \leftarrow s^\alpha)$ denotes the change in $E(\mathbf{s})$ due to changing $s_i$ to $s^\alpha$ and is given by:

$$\Delta E(s_i \leftarrow s^\alpha) = \frac{1}{2}\left[ \log \frac{\lambda_{a_i}(s_i)}{\lambda_{a_i}(s^{(\alpha)})} + \Delta s_i \lambda_{s_i} + \log(1 + \Delta\lambda_{a_i} J_{ii}) + \right.$$
$$\left. \frac{\Delta\lambda_{a_i}\hat{a}_i^2 - 2\hat{a}_i\Delta v_i - J_{ii}\Delta v_i^2}{1 + \Delta\lambda_{a_i} J_{ii}} + \Delta(\mu_i v_i) \right] \quad (19)$$

where $\Delta s_i = s^\alpha - s_i$, $\Delta\lambda_{a_i} = \lambda_{a_i}(s^\alpha) - \lambda_{a_i}(s_i)$, $\mathbf{J} = \mathbf{H}^{-1}$, and $v_i = \lambda_{a_i}(s_i)\,\mu_i(s_i)$. Note that all computations for considering a change of state are local and involve only terms with index $i$. Thus, deciding whether or not to change state can be computed quickly. However, if a change of state is accepted, then we must update $\mathbf{J}$. Using the Sherman-Morrison formula, this can be kept to an $O(N^2)$ computation:

$$\mathbf{J} \leftarrow \mathbf{J} - \left[\frac{\Delta\lambda_{a_k}}{1 + \Delta\lambda_{a_k} J_{kk}}\right] \mathbf{J}_k \mathbf{J}_k^T \quad (20)$$

As long as accepted state changes are rare (which we have found to be the case for sparse distributions), then Gibbs sampling may be performed quickly and efficiently. In addition, $\mathbf{H}$ and $\mathbf{J}$ are generally very sparse matrices, so as the system is scaled up the number of elements of $\mathbf{a}$ that are affected by a flip of $s_i$ will be relatively few.

In order to code images under this model, a single state of the coefficients must be chosen for a given image. We use for this purpose the MAP estimator:

$$\hat{\mathbf{a}} = \arg\max_{\mathbf{a}} P(\mathbf{a}|\mathbf{I}, \hat{\mathbf{s}}, \theta) \quad (21)$$

$$\hat{\mathbf{s}} = \arg\max_{\mathbf{s}} P(\mathbf{s}|\mathbf{I}, \theta) \quad (22)$$

Maximizing the posterior distribution over $\mathbf{s}$ is accomplished by assigning a temperature,

$$P(\mathbf{s}|\mathbf{I}, \theta) \propto e^{-E(\mathbf{s})/T} \quad (23)$$

and gradually lowering it until there are no more state changes.

# 5   Results

## 5.1   Test cases

We first trained the algorithm on a number of test cases containing known forms of both sparse and non-sparse (bi-modal) structure, using both critically sampled (complete) and 2×'s overcomplete basis sets. The training sets consisted of 6x6 pixel image patches that were created by a sparse superposition of basis functions (36 or 72) with $P(|s_i| = 1) = 0.2$, $\lambda_{a_i}(0) = 1000$, and $\lambda_{a_i}(1) = 10$. The results of these test cases confirm that the algorithm is capable of correctly extracting both sparse and non-sparse structure from data, and they are not shown here for lack of space.

## 5.2   Natural images

We trained the algorithm on 8x8 image patches extracted from pre-whitened natural images. In all cases, the basis functions were initialized to random functions (white noise) and the prior was initialized to be Gaussian (both Gaussians of roughly equal variance). Shown in figure 4a,b are the results for a set of 128 basis functions (2×'s overcomplete) in the two-Gausian case. In the three-Gaussian case, the prior was initialized to be platykurtic (all three Gaussians of equal variance but offset at three different positions). Thus, in this case the sparse form of the prior emerged completely from the data. The resulting priors for two of the coefficients are shown in figure 4c, with the posterior distribution averaged over many images overlaid. For some of the coefficients the posterior distribution matches the mixture-of-Gaussians prior well, but for others the tails appear more Laplacian in form. Also, it appears that the extra complexity offered by having three Gaussians is not utilized: Both Gaussians move to the center position and have about the same mean. When a non-sparse, bimodal prior is imposed, the basis function solution does not become localized, oriented, and bandpass as it does with sparse priors.

## 5.3   Coding efficiency

We evaluated the coding efficiency by quantizing the coefficients to different levels and calculating the total coefficient entropy as a function of the distortion introduced by quantization. This was done for basis sets containing 48, 64, 96, and 128 basis functions. At high SNR's the overcomplete basis sets yield better coding efficiency, despite the fact that there are more coefficients to code. However, the point at which this occurs appears to be well beyond the point where errors are no longer perceptually noticeable (around 14 dB).

# 6   Conclusions

We have shown here that both the prior and basis functions of our image model can be adapted to natural images. Without sparseness being imposed, the model both seeks distributions that are sparse and learns the appropriate basis functions for this distribution. Our conjecture that a small number of samples allows the posterior to be sufficiently characterized appears to hold. In all cases here, averages were collected over 40 Gibbs sweeps, with 10 sweeps for initialization. The algorithm proved capable of extracting the structure in challenging datasets in high dimensional spaces.

The overcomplete image codes have the lowest coding cost at high SNR levels, but at levels that appear higher than is practically useful. On the other hand, the

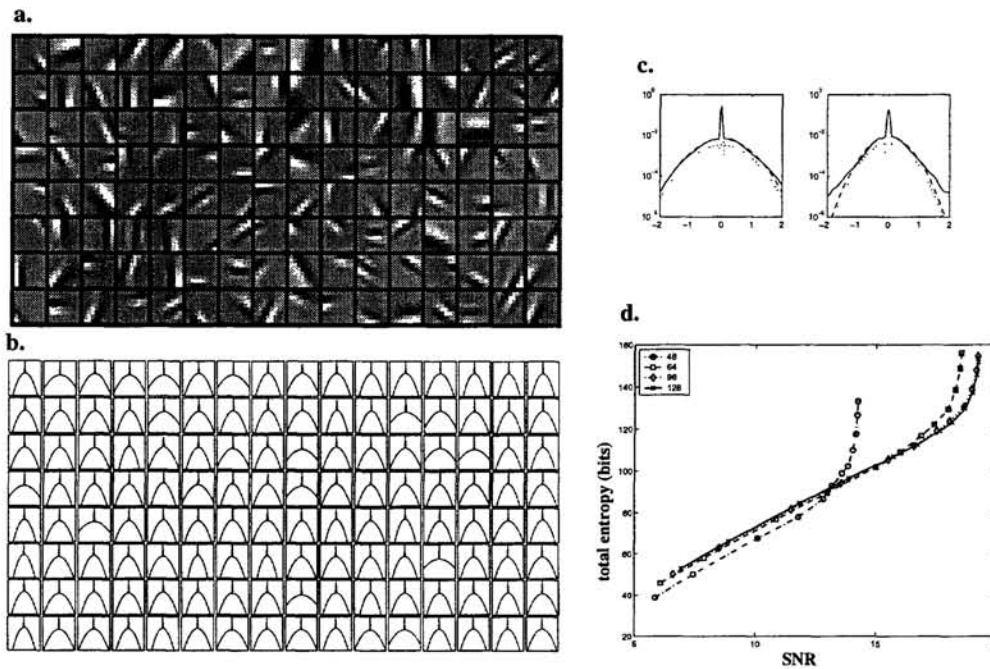

Figure 4: An overcomplete set of 128 basis functions (*a*) and priors (*b*, vertical axis is log-probability) learned from natural images. *c*, Two of the priors learned from a three-Gaussian mixture using 64 basis functions, with the posterior distribution averaged over many coefficients overlaid. *d*, Rate distortion curve comparing the coding efficiency of different learned basis sets.

sum of marginal entropies likely underestimates the true entropy of the coefficients considerably, as there are certainly statistical dependencies among the coefficients. So it may still be the case that the overcomplete bases will show a win at lower SNR's when these dependencies are included in the model (through the coupling term $\Lambda_s$).

## Acknowledgments

This work was supported by NIH grant R29-MH057921.

## References

[1] Olshausen BA, Field DJ (1997) "Sparse coding with an overcomplete basis set: A strategy employed by V1?" Vision Research, 37: 3311-3325.

[2] Bell AJ, Sejnowski TJ (1997) "The independent components of natural images are edge filters," Vision Research, 37: 3327-3338.

[3] van Hateren JH, van der Schaaff A (1997) "Independent component filters of natural images compared with simple cells in primary visual cortex," Proc. Royal Soc. Lond. B, 265: 359-366.

[4] Lewicki MS, Olshausen BA (1999) "A probabilistic framework for the adaptation and comparison of image codes," JOSA A, 16(7): 1587-1601.

[5] Simoncelli EP, Freeman WT, Adelson EH, Heeger DJ (1992) "Shiftable multiscale transforms," IEEE Transactions on Information Theory, 38(2): 587-607.

[6] Attias H (1999) "Independent factor analysis," Neural Computation, 11: 803-852.
